# Noise Characterization, Modeling, and Reduction for In Vivo Neural Recording

**Zhi Yang[1], Qi Zhao[2], Edward Keefer[3,4], and Wentai Liu[1]**
[1] University of California at Santa Cruz, [2] California Institute of Technology
[3] UT Southwestern Medical Center, [4] Plexon Inc
yangzhi@soe.ucsc.edu

## Abstract

Studying signal and noise properties of recorded neural data is critical in developing more efficient algorithms to recover the encoded information. Important issues exist in this research including the variant spectrum spans of neural spikes that make it difficult to choose a globally optimal bandpass filter. Also, multiple sources produce aggregated noise that deviates from the conventional white Gaussian noise. In this work, the spectrum variability of spikes is addressed, based on which the concept of adaptive bandpass filter that fits the spectrum of individual spikes is proposed. Multiple noise sources have been studied through analytical models as well as empirical measurements. The dominant noise source is identified as neuron noise followed by interface noise of the electrode. This suggests that major efforts to reduce noise from electronics are not well spent. The measured noise from in vivo experiments shows a family of $1/f^x$ spectrum that can be reduced using noise shaping techniques. In summary, the methods of adaptive bandpass filtering and noise shaping together result in several dB signal-to-noise ratio (SNR) enhancement.

## 1 Introduction

Neurons in the brain communicate through the firing of action potentials. This process induces brief "voltage" spikes in the surrounding environment that can be recorded by electrodes. The recorded neural signal may come from single, or multiple neurons. While single neurons only require a detection algorithm to identify the firings, multiple neurons require the separation of superimposed activities to obtain individual neuron firings. This procedure, also known as spike sorting, is more complex than what is required for single neurons.

Spike sorting has acquired general attention. Many algorithms have been reported in the literature [1–7], with each claiming an improved performance based on different data. Comparisons among different algorithms can be subjective and difficult. For example, benchmarks of in vivo recordings that thoroughly evaluate the performance of algorithms are unavailable. Also, synthesized sequences with benchmarks obtained through neuron models [8], isolated single neuron recordings [2], or simultaneous intra- and extra- cellular recordings [9] lack the in vivo recording environment. As a result, synthesized data provide useful but limited feedback on algorithms.

This paper discusses a noise study, based on which SNR enhancement techniques are proposed. These techniques are applicable to an unspecified spike sorting algorithm. Specifically, a procedure of online estimating both individual spike and noise spectrum is first applied. Based on the estimation, a bandpass filter that fits the spectrum of the underlying spike is selected. This maximally reduces the broad band noise without sacrificing the signal integrity. In addition, a comprehensive study of multiple noise sources are performed through lumped circuit model as well as measurements. Experiments suggest that the dominant noise is not from recording electronics,

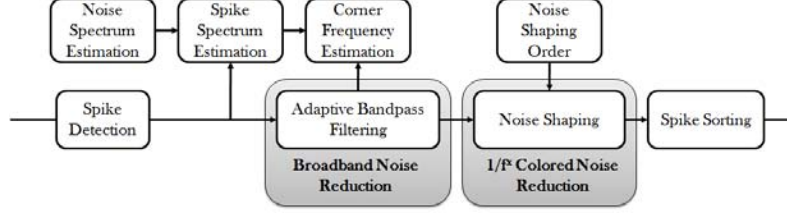

Figure 1: Block diagram of the proposed noise reduction procedures.

thus de-emphasize the importance of low noise hardware design. More importantly, the measured noise generally shows a family of $1/f^x$ spectrum, which can be reduced by using noise shaping techniques [10, 11]. Figure 1 shows the proposed noise reduction procedures.

The rest of this paper is organized as follows. Section 2 focuses on noise sources. Section 3 gives a Wiener kernel based adaptive bandpass filter. Section 4 describes a noise shaping technique that uses fractional order differentiation. Section 5 reports experiment results. Section 6 gives concluding remarks.

## 2   Noise Spectrum and Noise Model

Recorded neural spikes are superimposed with noise that exhibit non-Gaussian characteristics and can be approximated as $1/f^x$ noise. The frequency dependency of noise is contributed by multiple sources. Identified noise sources include $1/f^\alpha-$neuron noise [12–14] (notations of $1/f^x$ and $1/f^\alpha$ represent frequency dependencies of the total noise and neuron noise respectively), electrode-electrolyte interface noise [15], tissue thermal noise, and electronic noise, which are illustrated in Figure 2 using a lumped circuit model. Except electrolyte bulk noise ($4kTR_b$ in Figure 2) that has a flattened spectrum, the rest show frequency dependency. Specifically, $1/f^\alpha-$neuron noise is induced from distant neurons [12–14]. Numeric simulations based on simplified neuron models [12] suggest that $\alpha$ can vary a wide range depending on the parameters. For the electrode-electrolyte interface noise, non-faradaic type in particular, an effective resistance ($R_{ee}$) is defined for modeling purposes. $R_{ee}$ generates noise that is attenuated quadratically to frequency in high frequency region by the interface capacitance ($C_{ee}$). Electronic noise consists of two major components: thermal noise ($\sim kT/g_m$ [16]) and flicker noise (or $1/f$ noise [16]). Flicker noise dominates at lower frequency range and is fabrication process dependent. Next, we will address the noise model that will later be used to develop noise removal techniques in Section 3 and Section 4, and verified by experiment results in section 5.

### 2.1   $1/f^\alpha-$Neuron Noise

Background spiking activities of the vast distant neurons (e.g. spike, synaptic release [17–19]) overlap the spectrum of the recorded spike signal. They usually have small magnitudes and are noisily aggregated. Analytically, the background activities are described as

$$V_{neu} = \sum_i \sum_k v_{i.neu}(t - t_{i,k}), \tag{1}$$

where $V_{neu}$ represents the superimposed background activities of distant neurons; $i$ and $t_{i,k}$ represent the object identification and its activation time respectively, and $v_{i.neu}$ is the spiking activity template of the $i^{th}$ object. Based on Eq. 1, the power spectrum of $V_{neu}$ is

$$P\{V_{neu}\} = \sum_i \sum_k \frac{|X_i(f)|^2 f_i}{2} < e^{2\pi j f(t_{i,k_1+k} - t_{i,k_1})} >, \tag{2}$$

where $< >$ represents the average over the ensemble and over $k_1$, $P\{\}$ is the spectrum operation, $X_i(f)$ is the fourier transform of $v_{i.neu}$, and $f_i$ is the frequency of spiking activity $v_{i.neu}$ (the number of activations divided by a period of time). The spectrum of a delta function spike pulse

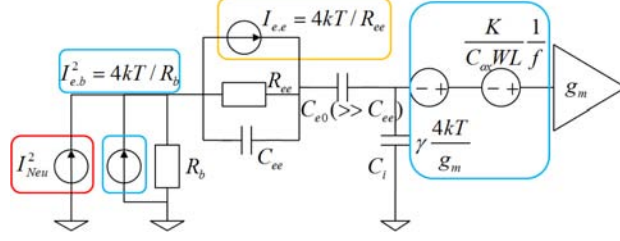

Figure 2: Noise illustration for extracellular spikes.

train ($\sum_k < e^{2\pi j f(t_{k_1 + k} - t_{k_1})} >$), according to [12], features a lower frequency and exhibits a $1/f^\alpha$ frequency dependency. As this term multiplies $|X_i(f)|^2$, the unresolved spiking activities of distant neurons contribute a spectrum of $1/f^x$ within the signal spectrum.

## 2.2 Electrode Noise

Assume the electrode-electrolyte interface is the non-faradaic type where charges such as electrons and ions, can not pass across the interface. In a typical in vivo recording environment that involves several different ionic particles, e.g. Na+, K+, ..., the current flux of any $i^{th}$ charged particle $J_i(x)$ at location $x$ assuming spatial concentration $n_i(x)$ is described by Nernst equation

$$J_i(x) = -D_i \nabla n_i(x) + n_i(x)\upsilon - \frac{z_i q}{kT} D_i n_i \nabla \Phi(x), \tag{3}$$

where $D_i$ is the diffusion coefficient, $\Phi$ electrical potential, $z_i$ charge of the particle, $q$ the charge of one electron, $k$ the Boltzmann constant, $T$ the temperature, and $\upsilon$ the convection coefficient. In a steady state, $J_i(x)$ is zero with the boundary condition of maintaining about $1V$ voltage drop from metal to electrolyte. In such a case, the electrode interface can be modeled as a lumped resistor $R_{ee}$ in parallel with a lumped capacitor $C_{ee}$. This naturally forms a lowpass filter for the interface noise. As a result, the induced noise from $R_{ee}$ at the input of the amplifier is

$$N_{e.e} = \frac{4kT}{R_{ee}}(R_{ee}||j\omega C_{ee}||(R_b + j\omega C_i))^2 = \frac{4kT}{R_{ee}}|\frac{1}{1/R_{ee} + j\omega C_{ee} + 1/(R_b + \frac{1}{j\omega C_i})}|^2. \tag{4}$$

Referring to the hypothesis that the amplifier input capacitance ($C_i$) is sufficiently small, introducing negligible waveform distortion, the integrated noise by electrode interface satisfies

$$\int_{f_{c1}}^{f_{c2}} N_{e.e}df \approx \int_{f_{c1}}^{f_{c2}} \frac{4kT R_{ee}}{|1 + 2\pi j f R_{ee} C_{ee}|^2} df = \frac{2kT}{\pi C_{ee}} tan^{-1} 2\pi R_{ee} C_{ee} f|_{f=f_{c1}}^{f=f_{c2}} < \frac{kT}{C_{ee}}. \tag{5}$$

Equation 5 suggests reducing electrode interface noise by increasing double layer capacitance ($C_{ee}$). Without increasing the size of electrodes, carbon-nanotube (CNT) coating [20] can dramatically increase electrode surface area, thus, reducing the interface noise. Section 5 will compare conventional electrodes and CNT coated electrodes from a noise point of view.

In regions away from the interface boundary, $\nabla n_i(x) = 0$ results in a flattened noise spectrum. Here we use a lumped bulk resistance $R_b$ in series with the double-layer interface for modeling noise

$$N_{e.b} = 4kT R_b = 4kT\chi\frac{\rho_{tissue}}{\pi r_s}, \tag{6}$$

where $R_b$ is the bulk resistance, $\rho_{tissue}$ is the electrolyte resistivity, $r_s$ is the radius of the electrode, and $\chi$ is a constant that relates to the electrode geometry. As given in [21], $\chi \approx 0.5$ for a plate electrode.

## 2.3 Electronic Noise

Noise generated by electronics can be predicted by circuit design tools and validated through measurements. At the frequency of interest, there are two major components: thermal noise of transistors and flicker noise

$$N_{electronic} = N_{c.thermal} + N_{c.flicker} = \gamma\frac{4kT}{g_m} + \frac{K}{C_{ox}WL}\frac{1}{f}, \tag{7}$$

where $N_{c.thermal}$ is the circuit thermal noise, $N_{c.flicker}$ the flicker noise, $g_m$ the transconductance of the amplifier ($\partial i_{out}/\partial v_{in}$), $\gamma$ a circuit architecture dependent constant on the order of $O(1)$, $K$ a process-dependent constant on the order of $10^{-25}V^2F$ [16], $C_{ox}$ the transistor gate capacitance density, and $W$ and $L$ the transistor width and length respectively.

Given a design schematic, circuit thermal noise can be reduced by increasing transconductance ($g_m$), which is to the first order linear to bias current thus power consumption. Flicker noise can be reduced using design techniques such as large size input transistors and chopper modulations [22]. By using advanced semiconductor technologies, also, power and area trade off to noise [16], and elegant design techniques like chopper modulation, current feedback [23], the state-of-the-art low noise neural amplifier can provide less than $2\mu V$ total noise [24]. Such design costs can be necessary and useful if electronics noise contributes significantly to the total noise. Otherwise, the over-designed noise specification may be used to trade off other specifications and potentially result in overall improved performance of the system. Section 5 will present experiments of evaluating noise contribution from different sources, which show that electronics are not the dominant noise source in our experiments.

## 2.4 Total Noise

The noise sources as shown in Figure 2 include unresolved neuron activities ($N_{neu}$), electrode-electrolyte interface noise ($N_{e.e}$), thermal noise from the electrolyte bulk ($N_{e.b}$) and active circuitry ($N_{c.thermal}$), and flicker noise ($N_{c.flicker}$). The noise spectrum is empirically fitted by

$$N(f) = N_{neu} + N_{e.e} + N_{e.b} + N_{c.thermal} + N_{c.flicker} \approx \frac{N_1}{f^x} + N_0, \qquad (8)$$

where $N_1/f^x$ and $N_0$ represent the frequency dependent and flat terms, respectively. Equation 8 describes a combination of both colored noise ($1/f^x$) and broad band noise, which can be reduced by using noise removal techniques. Section 3 presents an adaptive filtering scheme used to optimally attenuate the broad band noise. Section 4 presents a noise shaping technique used to improve the differentiation between signals and noise within the passband.

# 3 Adaptive Bandpass Filtering

SNR is calculated by integrating both signal and noise spectrum. Intuitively, a passband, either too narrow or wide, introduces signal distortion or unwanted noise. Figure 5(b) plots the detected spikes from one single electrode with different widths and shows the difficulty of optimally sizing the passband. While a passband that only fits one spike template may introduce waveform distortion to spikes of other templates, a passband that covers every template will introduce more noise to spikes of every template. A possible solution is to adaptively assign a passband to each spike waveform such that each span will be just wide enough to cover the underlying waveform. This section presents the steps used in order to achieve this solution and includes spike detection, spectrum estimation, and filter generation.

## 3.1 Spike Detection

In this work, spike detection is performed using a nonlinear energy operator (NEO) [25] that captures instantaneous high frequency and high magnitude activities. With a discrete time signal $x_i, i = ...1, 2, 3...$, NEO outputs

$$\psi(x_i) = x_i^2 - x_{i+1}x_{i-1}. \qquad (9)$$

The usefulness of NEO for spike detection can be explored by taking the expectation of Eq. 9

$$\overline{\psi(x_i)} = R_x(0) - R_x(2\triangle T) \approx \int P(f, \ i)(1 - cos4\pi f \triangle T)df, \qquad (10)$$

where $R_x$ is the auto correlation function, $\triangle T$ is the sampling interval, and $P(f, i)$ is the estimated power spectrum density with window centered at sample $x_i$. When the frequency of interest is much lower than the sampling frequency, $1 - cos2\pi f\tau$ is approximately $2\pi^2 f^2\tau^2$. This emphasizes the high frequency power spectrum. Because spikes are high frequency activities by definition, NEO outputs a larger score when spikes are present. An example of NEO based spike detection is shown in Figure 4, where NEO improves the separation between spikes and the background activity.

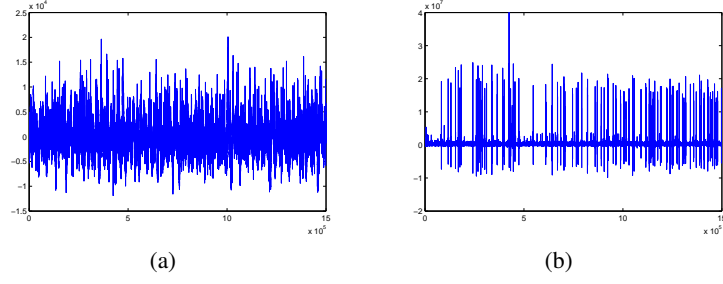

(a)                       (b)

Figure 3: Spike sequence and its corresponding NEO output. (a) Raw sequence of one channel. (b) The corresponding NEO output of the raw sequence in (a).

### 3.2    Corner Frequency Estimation

Spectrum estimation of individual spikes is performed to select a corresponding bandpass filter that balances the spectrum distortion and noise. Knowing its ability to separate bandlimited signals from broad band noise, a Weiner filter [26] is used here to size the signal passband. In the frequency domain, denoting $P_{XX}$ and $P_{NN}$ as the signal and noise spectra, Weiner filter is

$$W(f) = \frac{P_{XX}(f)}{P_{XX}(f) + P_{NN}(f)} = \frac{SNR(f)}{SNR(f) + 1}. \tag{11}$$

Implementing a precise Weiner filter for each detected spike requires considerable computation, as well as a reliable estimation of the signal spectrum. In this work, we are interested in using one of a series of prepared bandpass filters $H_i$ $(i = 1, 2...n)$ that better matches the solved "optimal" Weiner filter

$$\arg\min_i \int |H_i(f) - W(f)|^2 df, \tag{12}$$

subjected to $\int [H_i(f) - W(f)] df = 0$.

## 4    Noise Shaping

The adaptive scheme presented in Section 3 tacitly assigns a matched frequency mask to individual spikes and balances noise and spectrum integrity. The remaining noise exhibits $1/f^x$ frequency dependency according to Section 2. In this section, we focus on noise shaping techniques to further distinguish signal from noise.

The fundamentals of noise shaping are straightforward. Instead of equally amplifying the spectrum, a noise shaping filter allocates more weight to high SNR regions while reducing weight at low SNR regions. This results in an increased ratio of the integrated signal power over the noise power. In general, there are a variety of noise shaping filters that can improve the integrated SNR [10]. In this work, we use a series of fractional derivative operation for noise shaping

$$D(h(x)) = \frac{d^p h(x)}{dx^p}, \tag{13}$$

where $h(x)$ is a general function, $p$ is a positive number (can be integer or non-integer) that adjusts the degree of noise shaping; the larger the $p$, the more emphasis on high frequency spectrum. In $Z$ domain, the realization of fractional derivative operation can be done using binomial series [27]

$$H(z) = (1 - z^{-1})^p = \sum_{n=0}^{\infty} h(n) z^{-k} = 1 - pz^{-1} + \sum_{n=2}^{\infty} (-1)^n \frac{p(p-1)...(p-n+1)}{n!} z^{-n}, \tag{14}$$

where $h(n)$ are the fractional derivative filter coefficients that converge to zero.

The SNR gain in applying a fractional derivative filter $H(f)$ is

$$SNR_{gain} = 10 log \frac{\int I_{spike}(f)|H(f)|^2 df}{\int I_{noise}(f)|H(f)|^2 df} - 10 log \frac{\int I_{spike}(f) df}{\int I_{noise}(f) df}, \tag{15}$$

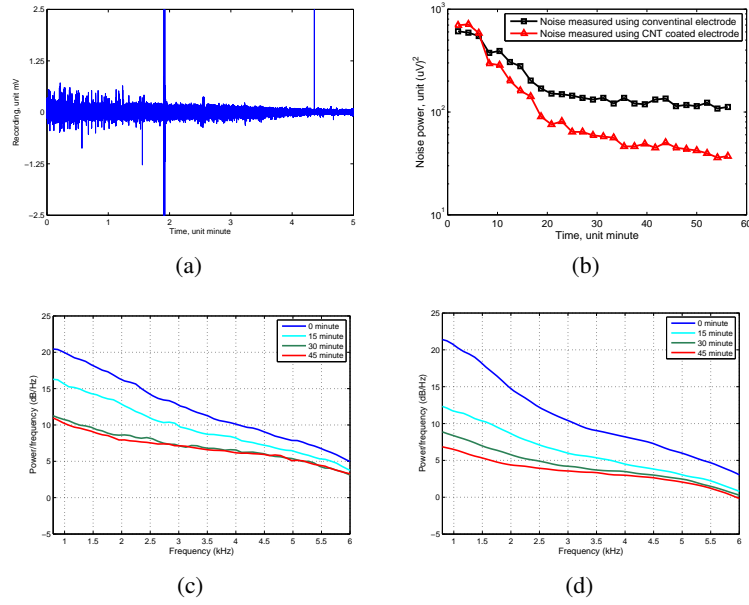

Figure 4: In vivo recording for identifying noise sources. (a) 5-minute recording segment capturing the decay of background activities. (b) Traces of the estimated noise vs. time are plotted. Black ■ curve represents the noise recorded from a custom tungsten electrode; red ▲ curve represents the noise recorded from a CNT coated electrodes with the same size. (c), (d) Noise power spectrums estimated at the 0, 15, 30, 45 minutes after the drug injection. In (c) a conventional tungsten electrode is used. In (d), a CNT coated tungsten electrode of equal size is used for comparison.

where $I_{spike}(f)$ and $I_{noise}(f)$ are power spectrums of spike and noise respectively. Numeric values of SNR gain depend on both data and $p$ (the degree of noise shaping). In our experiments, we empirically choose $p$ in a range of $0.5$ to $2.5$, where numerically calculated SNR gains using Eq. 15 of in vivo recordings are typically more than 3dB, which is consistent with [10].

# 5  Experiment

To verify the noise analysis presented in Section 2, an in vivo experiment is performed that uses two sharp tungsten electrodes separated by 125 $\mu m$ to record the hippocampus neuronal activities of a rat. One of the electrodes is coated with carbon-nanotube (CNT), while the other is uncoated. After the electrodes have been placed, a euthanizing drug is injected. After 5 seconds of drug injection, the recording of the two electrodes start and last until to the time of death. The noise analysis results are summarized and presented in Figure 4. In Figure 4(a), a 5-minute segment that captures the decaying of background activities is plotted. In Figure 4(b), the estimated noise from $600Hz$ to $6KHz$ for both recording sites are plotted, where noise dramatically reduces ($> 80\%$) after the drug takes effect. Initially, the CNT electrode records a comparatively larger noise ($697\mu V^2$) compared with the uncoated electrode ($610\mu V^2$). After a few minutes, the background noise recorded by the CNT electrode quickly reduces eventually reaching $37\mu V^2$ that is about $1/3$ of noise recorded by its counterpart ($112\mu V^2$), suggesting the noise floor of using the uncoated tungsten electrode ($112\mu V^2$) is set by the electrode. From these two plots, we can estimate that the neuron noise is around $500 \sim 600\mu V^2$, electrode interface noise is $\sim 80\mu V$, while the sum of electronic noise and electrolyte bulk noise is less than $37\mu V^2$ (only $\sim 5\%$ of the total noise). Figure 4(c) displays the $1/f^x$ noise spectrum recorded from the uncoated tungsten electrode ($x = 1.8, 1.4, 1.0, 0.9$, estimated at $0, 15, 30, 45$ minutes after drug injection). Figure 4(d) displays $1/f^x$ noise spectrum recorded from the CNT coated electrode ($x = 2.1, 1.3, 0.9, 0.8$, estimated at $0, 15, 30, 45$ minutes after drug injection).

Table 1: Statistics of $1/f^x$ noise spectrum from in vivo preparations.

| $1/f^x$ | $x < 1$ | $1 \leq x < 1.5$ | $1.5 \leq x < 2$ | $x \geq 2$ |
|---|---|---|---|---|
| Number of Recordings | 5 | 38 | 23 | 11 |

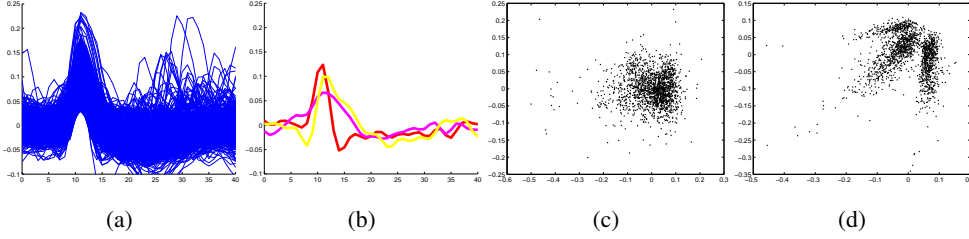

| (a) | (b) | (c) | (d) |

Figure 5: In vivo experiment of evaluating the proposed adaptive bandpass filter. (a) Detected spikes are aligned and superimposed. (b) Example waveforms that have distinguished widths are plotted. (c) Feature extraction results using PCA with a global bandpass filter ($400Hz$ to $5KHz$) are displayed. (d) Feature extraction results using PCA with adaptive bandpass filters are displayed showing a much improved cluster isolation compared to (c).

In the second experiment, 77 recordings of in vivo preparations are used to explore the stochastic distribution of $1/f^x$ noise spectrum. "$x$" is averaged at $1.5$ with a standard deviation of $0.5$ ($1/f^{1.5 \pm 0.5}$). The results are summarized in Table 1.

The third experiment uses an in vivo recording from a behaving cat. This recording is used to compare the feature extraction results produced by a global bandpass filter (conventional one) and the proposed adaptive bandpass filter, discussed in Section 3. In Figure 5(a), detected spikes are superimposed, where "a thick waveform bundle" is observed. In Figure 5(b), example waveforms in Figure 5(a) that have different widths are shown. Clearly, these waveforms have noticeably different spectrum spans. In Figure 5(c), feature extraction results using PCA (a widely used feature extraction algorithm in spike sorting applications) with a global bandpass filter are displayed. As a comparison, feature extraction results using PCA with adaptive bandpass filters are displayed in Figure 5(d), where multiple clusters are differentiable in the feature space.

In the fourth experiment, earth mover's distance (EMD), as a cross-bin similarity measure that is robust to waveform misalignment [28], is applied to synthesized data for evaluation of the spike waveform separation before and after noise shaping. Assume $V_A(i), i = 1, 2..., V_B(i), i = 1, 2...$ to be the spike waveform bundles from candidate neuron $A$ and $B$. To estimate the spike variation of a candidate neuron, two waveforms are randomly picked from a same waveform bundle, and the distance between them is calculated using EMD. After repeating the procedure many times, the results are plotted as the black (waveforms from $V_A$) and blue (waveforms from $V_A$) traces in Figure 6. The $x$-axis indexes the trial and the $y$-axis is the EMD score. Black/blue traces describe the intra-cluster waveform variations of the two neurons under testing. To estimate the separation between candidate neuron $A$ and $B$, we randomly pick two waveforms, one from $V_A$ and the other from $V_B$, then compute the EMD between them. This procedure is repeated many times and the EMD vs. trial index is plotted as the red curve in Figure 6. Four pairs of candidate neurons are tested and shown in Figure 6(a)-(d). It can be observed from Figure 6 that the red curves are not well differentiated from the black/blue ones, which indicate that candidate neurons are not well separated. In Figure 6(e)-(h), we apply a similar procedure on the same four pairs of candidate neurons. The only difference from plots shown in Figure 6(a)-(d) is that the waveforms after noise shaping are used rather than their original counterparts. In Figure 6(e)-(h), the red curves separate from the black/blue traces, suggesting that the noise shaping filter improves waveform differentiations.

In the fifth experiment, we apply different orders of noise shaping filters and the same feature extraction algorithm to evaluate the feature extraction results. The noise shaping technique is developed as a general tool that can be incorporated into an unspecified feature extraction algorithm. Here, we use PCA as an example. In Figure 7, 8 figures in the same row are results of the same sequence. Figures from left to right display the feature extraction results with different orders of noise shaping; from 0 (no noise shaping) to 3.5, and stepped by 0.5. All the tests are obtained after adaptive band-

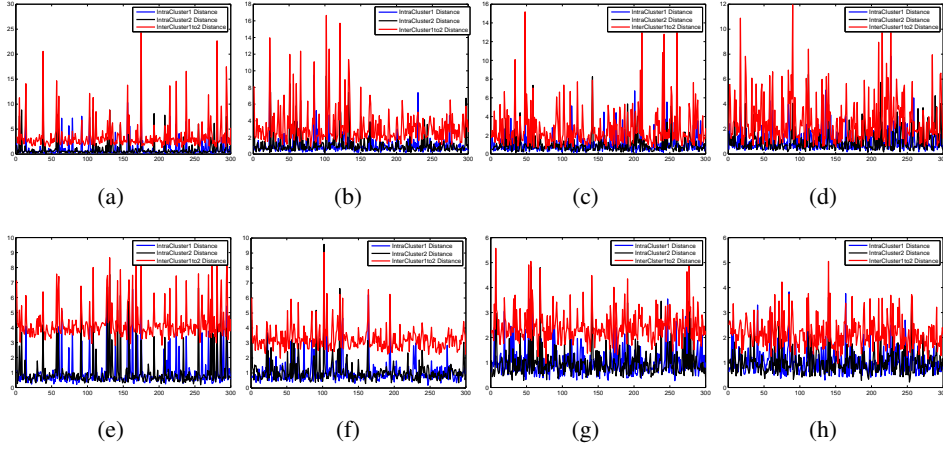

(a)          (b)          (c)          (d)

(e)          (f)          (g)          (h)

Figure 6: EMD vs. trial index. Black and blue trace: EMDs for intra-cluster waveforms; red trace: EMDs for inter-cluster waveforms. (a)-(d) and (e)-(h) are results of 4 different pairs of neurons before and after noise shaping respectively. Traces in (a)-(d) and (e)-(h) have one-to-one correspondence. Noise level increases from (a) to (d).

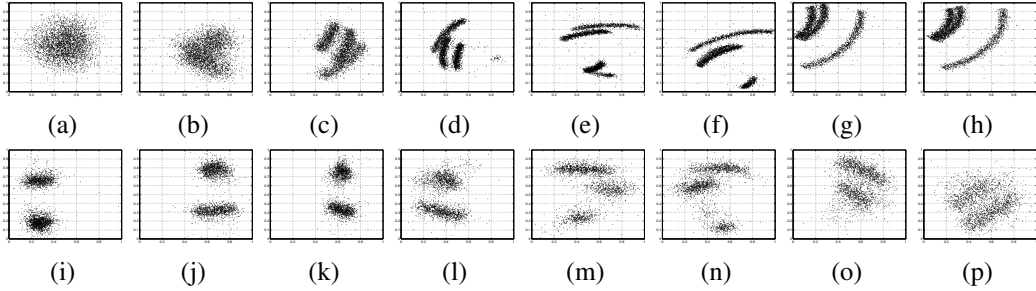

(a)     (b)     (c)     (d)     (e)     (f)     (g)     (h)

(i)     (j)     (k)     (l)     (m)     (n)     (o)     (p)

Figure 7: Feature extraction results using PCA with different orders of noise shaping. Each row represents a different sequence. Each column represents a different order of noise shaping ($p$ in $\frac{d^p f(x)}{dx^p}$), sweeping from 0 (without noise shaping) to 3.5 , stepped by 0.5. (a)-(h) are results of a synthesized sequence. (i)-(p) are results of an in vivo preparation. Clearly, (f) is better than (a); (m) is better than (i).

pass filtering. The first sequence (a)-(h) is a synthesized one from public data base [2], the second sequence is recorded from an in vivo preparation. For both sequences, increased numbers of isolated clusters can be obtained by appropriately choosing the order of the noise shaping filter.

## 6   Conclusion

In this paper, a study of multiple noise sources for in vivo neural recording is carried out. The dominant noise source is identified to be neuron noise followed by interface noise of the electrode. Overall, the noise exhibits a family of $1/f^x$ spectrum. The concept of adaptive bandpass filter is proposed to reduce noise because it maintains the signal spectrum integrity while maximally reducing the broad band noise. To reduce the noise within the signal passband and improve waveform separation, a series of fractional order differentiator based noise shaping filters are proposed. The proposed noise removal techniques are generally applicable to an unspecified spike sorting algorithm. Experiment results from in vivo preparations, synthesized sequences, and comparative recordings using both conventional and CNT coated electrodes are reported, which verify the noise model and demonstrate the usefulness of the proposed noise removal techniques.

# References

[1] Lewicki MS. A review of methods for spike sorting: the detection and classification of neural action potentials. Network Comput Neural Syst. 1998;9:53–78.

[2] Quian Quiroga R, Nadasdy Z, Ben-Shaul Y. Unsupervised spike detection and sorting with wavelets and superparamagnetic clustering. Neural Computation. 2004 Aug;16(8):1661–1687.

[3] Bar-Hillel A, Spiro A, Stark E. Spike sorting: Bayesian clustering of non-stationary data. Advances in Neural Information Processing Systems 17. 2005;p. 105–112.

[4] Zumsteg ZS, Kemere C, O'Driscoll S, Santhanam G, Ahmed RE, Shenoy KV, et al. Power feasibility of implantable digital spike sorting circuits for neural prosthetic systems. IEEE Trans Neural Syst Rehabil Eng. 2005 Sep;13(3):272–279.

[5] Vargas-Irwin C, Donoghue JP. Automated spike sorting using density grid contour clustering and subtractive waveform decomposition. J Neurosci Methods. 2007;164(1).

[6] Yang Z, Zhao Q, Liu W. Neural signal classification using a simplified feature set with nonparametric clustering. Neurocomputing, doi:101016/jneucom200907013, in press;.

[7] Gasthaus J, Wood F, Dilan G, Teh YW. Dependent dirichlet process spike sorting. Advances in Neural Information Processing Systems 21. 2009;p. 497–504.

[8] Smith LS, Mtetwa N. A tool for synthesizing spike trains with realistic interference. J Neurosci Methods. 2007 Jan;159(1):170–180.

[9] Harris KD, Henze DA, Csicsvari J, Hirase H, Buzsaki G. Accuracy of tetrode spike separation as determined by simultaneous intracellular and extracellular measurements. J Neurophysiol. 2000;84:401–414.

[10] Yang Z, Zhao Q, Liu W. Spike feature extraction using informative samples. Advances in Neural Information Processing Systems 21. 2009;p. 1865–1872.

[11] Yang Z, Zhao Q, Liu W. Improving Spike Separation Using Waveform Derivative. Journal of Neural Engineering, doi: 101088/1741-2560/6/4/046006. 2009 Aug;6(4).

[12] Davidsen J, Schuster HZ. Simple model for $1/f^\alpha$ noise. Phys Rev Lett 65, 026120(1)-026120(4). 2002;.

[13] Yu Y, Romero R, Lee TS. Preference of sensory neural coding for 1/f signals. Phys Rev Lett 94, 108103(1)-108103(4). 2005;.

[14] Bedard C, Kroger H, Destexhe A. Does the 1/f frequency scaling of brain reflect self-organized critical states? Phys Rev Lett 97, 118102(1)-118102(4). 2006;.

[15] Hassibi A, Navid R, Dutton RW, Lee TH. Comprehensive study of noise processes in electrode electrolyte interfaces. J Appl Phys. 2004 July;96(2):1074–1082.

[16] Razavi B. Design of Analog CMOS Integrated Circuits. Boston, MA:McGraw-Hill; 2001.

[17] Keener J, Sneyd J. Mathematical Physiology. New York: Springer Verlag; 1998.

[18] Manwani A, N Steinmetz P, Koch C. Channel noise in excitable neural membranes. Advances in Neural Information Processing Systems 12. 2000;p. 142–149.

[19] Fall C, Marland E, Wagner J, Tyson J. Computational Cell Biology. New York: Springer Verlag; 2002.

[20] Keefer EW, Botterman BR, Romero MI, Rossi AF, Gross GW. Carbon nanotube-coated electrodes improve brain readouts. Nat Nanotech. 2008;3:434–439.

[21] Wiley JD, Webster JG. Analysis and control of the current distribution under circular dispersive. IEEE Trans Biomed Eng. 1982;29:381–385.

[22] Denison T, Consoer K, Kelly A, Hachenburg A, Santa W. A $2.2\mu$W $94nV/\sqrt{Hz}$, chopper-stabilized instrumentation amplifier for EEG detection in chronic implants. IEEE ISSCC Dig Tech Papers. 2007 Feb;8(6).

[23] Ferrari G, Gozzini F, Sampietro M. A current-sensitive front-end amplifier for nano biosensors with a 2MHz BW. IEEE ISSCC Dig Tech Papers. 2007 Feb;8(7).

[24] Harrison RR. The design of integrated circuits to observe brain activity. Proc IEEE. 2008 July;96:1203–1216.

[25] Kaiser JF. On a simple algorithm to calculate the energy of a signal. In Proc IEEE Int Conf Acoustic Speech and Signal Processing. 1990;p. 381–384.

[26] Vaseghi SV. Advanced Digital Signal Processing and Noise Reduction. Wiley-Teubner; 1996.

[27] Hosking J. Fractional differencing. Biometrika. 1981 Jan;68:165–176.

[28] Rubner Y. Perceptual metrics for image database navigation. In: Ph.D. dissertation, Stanford University; 1999. .

